# Grouping with Bias

**Stella X. Yu**
Robotics Institute
Carnegie Mellon University
Center for the Neural Basis of Cognition
Pittsburgh, PA 15213-3890
*stella.yu@cs.cmu.edu*

**Jianbo Shi**
Robotics Institute
Carnegie Mellon University
5000 Forbes Ave
Pittsburgh, PA 15213-3890
*jshi@cs.cmu.edu*

## Abstract

With the optimization of pattern discrimination as a goal, graph partitioning approaches often lack the capability to integrate prior knowledge to guide grouping. In this paper, we consider priors from unitary generative models, partially labeled data and spatial attention. These priors are modelled as constraints in the solution space. By imposing uniformity condition on the constraints, we restrict the feasible space to one of smooth solutions. A subspace projection method is developed to solve this constrained eigenproblem. We demonstrate that simple priors can greatly improve image segmentation results.

## 1 Introduction

Grouping is often thought of as the process of finding intrinsic clusters or group structures within a data set. In image segmentation, it means finding objects or object segments by clustering pixels and segregating them from background. It is often considered a bottom-up process. Although never explicitly stated, higher level of knowledge, such as familiar object shapes, is to be used only in a separate post-processing step.

The need for the integration of prior knowledge arises in a number of applications. In computer vision, we would like image segmentation to correspond directly to object segmentation. In data clustering, if users provide a few examples of clusters, we would like a system to adapt the grouping process to achieve the desired properties. In this case, there is an intimate connection to learning classification with partially labeled data.

We show in this paper that it is possible to integrate both bottom-up and top-down information in a single grouping process. In the proposed method, the bottom-up grouping process is modelled as a graph partitioning [1, 4, 12, 11, 14, 15] problem, and the top-down knowledge is encoded as constraints on the solution space. Though we consider normalized cuts criteria in particular, similar derivation can be developed for other graph partitioning criteria as well. We show that it leads to a constrained eigenvalue problem, where the global optimal solution can be obtained by eigendecomposition. Our model is expanded in detail in Section 2. Results and conclusions are given in Section 3.

## 2 Model

In graph theoretic methods for grouping, a relational graph $G_A = (V, E, W)$ is first constructed based on *pairwise* similarity between two elements. Associated with the graph edge between vertex $i$ and $j$ is weight $W_{ij}$, characterizing their likelihood of belonging in the same group.

For image segmentation, pixels are taken as graph nodes, and pairwise pixel similarity can be evaluated based on a number of low level grouping cues. Fig. 1c shows one possible definition, where the weight between two pixels is inversely proportional to the magnitude of the strongest intervening edge [9].

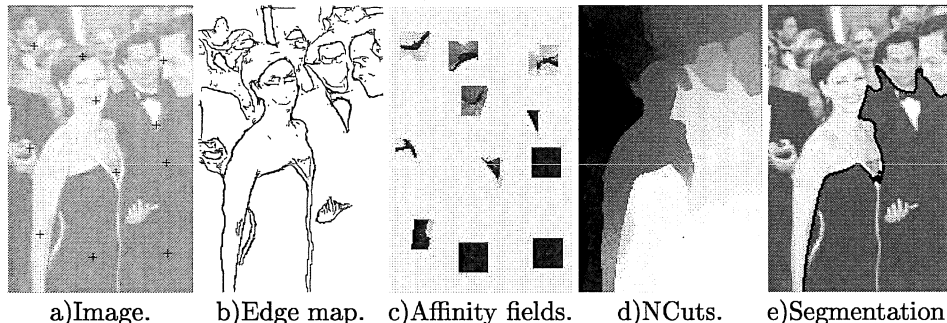

| a)Image. | b)Edge map. | c)Affinity fields. | d)NCuts. | e)Segmentation. |

Figure 1: Segmentation by graph partitioning. a)200 × 129 image with a few pixels marked(+). b)Edge map extracted using quadrature filters. c)Local affinity fields of marked pixels superimposed together. For every marked pixel, we compute its affinity with its neighbours within a radius of 10. The value is determined by a Gaussian function of the maximum magnitude of edges crossing the straight line connecting the two pixels [9]. When there is a strong edge separating the two, the affinity is low. Darker intensities mean larger values. d)Solution by graph partitioning. It is the second eigenvector from normalized cuts [15] on the affinity matrix. It assigns a value to each pixel. Pixels of similar values belong to the same group. e)Segmentation by thresholding the eigenvector with 0. This gives a bipartitioning of the image which corresponds to the best cuts that have maximum within-region coherence and between-region distinction.

After an image is transcribed into a graph, image segmentation becomes a vertex partitioning problem. Consider segmenting an image into foreground and background. This corresponds to vertex bipartitioning $(V_1, V_2)$ on graph $G$, where $V = V_1 \cup V_2$ and $V_1 \cap V_2 = \varnothing$. A good segmentation seeks a partitioning such that nodes within partitions are tightly connected and nodes across partitions are loosely connected. A number of criteria have been proposed to achieve this goal. For normalized cuts [15], the solution is given by some eigenvector of weight matrix $W$ (Fig. 1d). Thresholding on it leads to a discrete segmentation (Fig. 1e). While we will focus on normalized cuts criteria [15], most of the following discussions apply to other criteria as well.

### 2.1 Biased grouping as constrained optimization

Knowledge other than the image itself can greatly change the segmentation we might obtain based on such low level cues. Rather than seeing boundaries between black and white regions, we see objects. The sources of priors we consider in this paper are: unitary generative models (Fig. 2a), which could arise from sensor models in MRF [5], partial grouping (Fig. 2b), which could arise from human computer interaction [8], and spatial attention (Fig. 2c). All of these provide additional, often long-range, binding information for grouping.

We model such prior knowledge in the form of constraints on a valid grouping configuration. In particular, we see that all such prior knowledge defines a partial

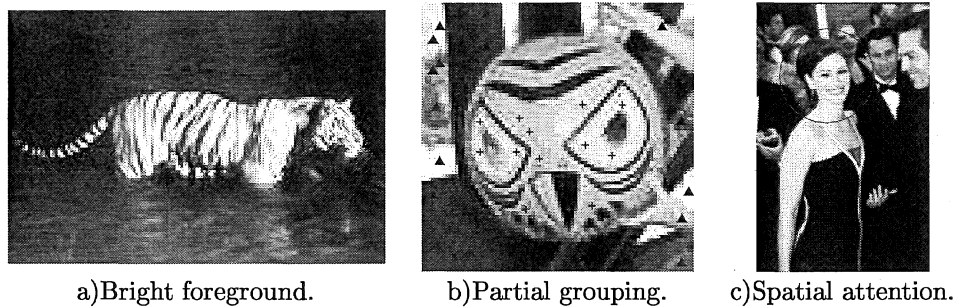

| a)Bright foreground. | b)Partial grouping. | c)Spatial attention. |

Figure 2: Examples of priors considered in this paper. a)Local constraints from unitary generative models. In this case, pixels of light(dark) intensities are likely to be the foreground(background). This prior knowledge is helpful not only for identifying the tiger as the foreground, but also for perceiving the river as one piece. How can we incorporate these unitary constraints into a graph that handles only pairwise relationships between pixels? b)Global configuration constraints from partial grouping *a priori*. In this case, we have manually selected two sets of pixels to be grouped together in foreground (+) and background (▲) respectively. They are distributed across the image and often have distinct local features. How can we force them to be in the same group and further bring similar pixels along and push dissimilar pixels apart? c)Global constraints from spatial attention. We move our eyes to the place of most interest and then devote our limited visual processing to it. The complicated scene structures in the periphery can thus be ignored while sparing the parts associated with the object at fovea. How can we use this information to facilitate figural popout in segmentation?

grouping solution, indicating which set of pixels should belong to one partition. Let $H_l$, $l = 1, \cdots, n$, denote a partial grouping. $H_t$ have pixels known to be in $V_t, t = 1, 2$. These sets are derived as follows.

*Unitary generative models:* $H_1$ and $H_2$ contains a set of pixels that satisfy the unitary generative models for foreground and background respectively. For example, in Fig. 2a, $H_1(H_2)$ contains pixels of brightest(darkest) intensities.

*Partial grouping:* Each $H_l$, $l = 1, \cdots, n$, contains a set of pixels that users specify to belong together. The relationships between $H_l$, $l > 2$ and $V_t$, $t = 1, 2$ are indefinite.

*Spatial attention:* $H_1 = \varnothing$ and $H_2$ contains pixels randomly selected outside the visual fovea, since we want to maintain maximum discrimination at the fovea but merging pixels far away from the fovea to be one group.

To formulate these constraints induced on the graph partitioning, we introduce binary group indicators $X = [X_1, X_2]$. Let $N = |V|$ be the number of nodes in the graph. For $t = 1, 2$, $X_t$ is an $N \times 1$ vector where $X_t(k) = 1$ if vertex $k \in V_t$ and 0 otherwise. The constrained grouping problem can be formally written as:

$$\begin{aligned} \min \quad & \epsilon(X_1, X_2) \\ \text{s.t.} \quad & X_t(i) = X_t(j), \ i, j \in H_l, \ l = 1, \cdots, n, \ t = 1, 2, \\ & X_t(i) \neq X_t(j), \ i \in H_1, \ j \in H_2, \ t = 1, 2, \end{aligned}$$

where $\epsilon(X_1, X_2)$ is some graph partitioning cost function, such as minimum cuts [6], average cuts [7], or normalized cuts [15]. The first set of constraints can be re-written in matrix form: $U^T X = 0$ , where, e.g. for some column $k$, $U_{ik} = 1$, $U_{jk} = -1$. We search for the optimal solution only in the feasible set determined by all the constraints.

## 2.2   Conditions on grouping constraints

The above formulation can be implemented by the maximum-flow algorithm for minimum cuts criteria [6, 13, 3], where two special nodes called source and sink are

introduced, with infinite weights set up between nodes in $H_1(H_2)$ and source(sink). In the context of learning from labeled and unlabeled data, the biased mincuts are linked to minimizing leave-one-out cross validation [2]. In the normalize cuts formulation, this leads to a constrained eigenvalue problem, as soon to be seen.

However, simply forcing a few nodes to be in the same group can produce some undesirable graph partitioning results, illustrated in Fig. 3. Without bias, the data points are naturally first organized into top and bottom groups, and then subdivided into left and right halves (Fig. 3a). When we assign points from top and bottom clusters to be together, we do not just want one of the groups to lose its labeled point to the other group (Fig. 3b), but rather we desire the biased grouping process to explore their neighbouring connections and change the organization to left and right division accordingly.

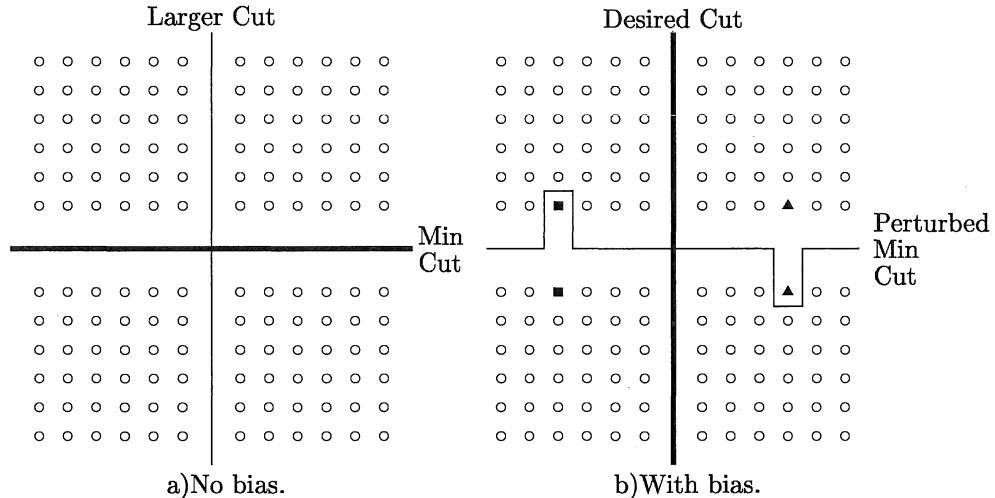

Figure 3: Undesired grouping caused by simple grouping constraints. a)Data points are distributed in four groups, with a larger spatial gap between top and bottom groups than that between left and right groups. Defining weights based on proximity, we find the top-bottom grouping as the optimal bisection. b)Introduce two pairs of filled nodes to be together. Each pair has one point from the top and the other from the bottom group. The desired partitioning should now be the left-right division. However, perturbation on the unconstrained optimal cut can lead to a partitioning that satisfies the constraints while producing the smallest cut cost.

The desire of propagating partial grouping information on the constrained nodes is, however, not reflected in the constrained partitioning criterion itself. Often, a slightly perturbed version of the optimal unbiased cut becomes the legitimate optimum. One reason for such a solution being undesirable is that some of the "perturbed" nodes are isolated from their close neighbours.

To fix this problem, we introduce the notion of *uniformity* of a graph partitioning. Intuitively, if two labeled nodes, $i$ and $j$, have similar connections to their neighbours, we desire a cut to treat them fairly so that if $i$ gets grouped with $i$'s friends, $j$ also gets grouped with $j$'s friends (Fig. 3b). This uniformity condition is one way to propagate prior grouping information from labeled nodes to their neighbours.

For normalized cuts criteria, we define the normalized cuts of a *single node* to be

$$\text{NCuts}(i; X) = \frac{\sum_{X_t(k) \neq X_t(i), \forall t} W_{ik}}{D_{ii}}.$$

This value is high for a node isolated from its close neighbours in partitioning $X$.

We may not know in advance what this value is for the optimal partitioning, but we desire this value to be the same for any pair of nodes preassigned together:

$$\text{NCuts}(i; X) = \text{NCuts}(j; X), \forall i, j \in H_l, \, l = 1, \cdots, n.$$

While this condition does not force $NCuts(i; X)$ to be small for each labeled node, it is unlikely for all of them to have a large value while producing the minimum NCuts for the global bisection. Similar measures can be defined for other criteria. In Fig. 4, we show that the uniformity condition on the bias helps preserving the smoothness of solutions at every labeled point. Such smoothing is necessary especially when partially labeled data are scarce.

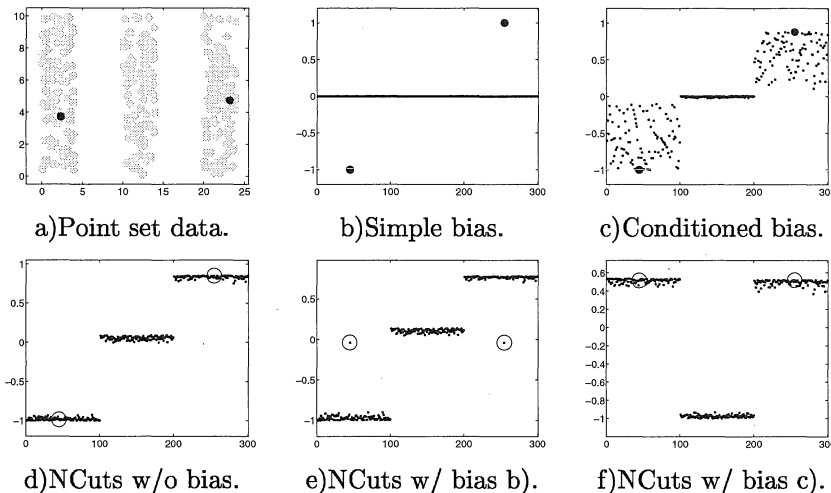

a)Point set data.      b)Simple bias.      c)Conditioned bias.

d)NCuts w/o bias.      e)NCuts w/ bias b).      f)NCuts w/ bias c).

Figure 4: Condition constraints with uniformity. a)Data consist of three strips, with 100 points each, numbered from left to right. Two points from the side strips are randomly chosen to be pre-assigned together. b)Simple constraint $U^T X = 0$ forces any feasible solution to have equal valuation on the two marked points. c)Conditioned constraint $U^T P X = 0$. Note that now we cannot tell which points are biased. We compute $W$ using Gaussian function of distance with $\sigma = 3$. d)Segmentation without bias gives three separate groups. e)Segmentation with simple bias not only fails to glue the two side strips into one, but also has two marked points isolated from their neighbours. f)Segmentaion with conditioned bias brings two side strips into one group. See the definition of $P$ below.

## 2.3 Computation: subspace projection

To develop a computational solution for the constrained optimization problem, we introduce some notations. Let the degree matrix $D$ be a diagonal matrix, $D_{ii} = \sum_k W_{ik}, \forall i$. Let $P = D^{-1}W$ be the normalized weight matrix. It is a transition probability matrix for nonnegative weight matrix $W$ [10]. Let $\alpha = \frac{X_1^T D X_1}{1^T D 1}$ be the degree ratio of $V_1$, where 1 is the vector of ones. We define a new variable $x = (1 - \alpha)X_1 - \alpha X_2$. We can show that for normalized cuts, the biased grouping with the *uniformity* condition is translated into:

$$\min \, \epsilon(x) = \frac{x^T(D - W)x}{x^T D x}, \text{ s.t. } U^T P x = 0.$$

Note, we have dropped the constraint $X_t(i) \neq X_t(j)$, $i \in H_1$, $j \in H_2$, $t = 1, 2$.

Using Lagrange multipliers, we find that the optimal solution $x^*$ satisfies:

$$QPx^* = \lambda x^*, \quad \epsilon(x^*) = 1 - \lambda,$$

where $Q$ is a projector onto the feasible solution space:

$$Q = I - D^{-1}V(V^T D^{-1}V)^{-1}V^T, \quad V = P^T U.$$

Here we assume that the conditioned constraint $V$ is of full rank, thus $V^T D^{-1}V$ is invertible. Since 1 is still the trivial solution corresponding to the largest eigenvalue of 1, the second leading right eigenvector of the matrix $QP$ is the solution we seek.

To summarize, given weight matrix $W$, partial grouping in matrix form $U^T x = 0$, we do the following to find the optimal bipartitioning:

Step 1: Compute degree matrix $D$, $D_{ii} = \sum_j W_{ij}$, $\forall i$.
Step 2: Compute normalized matrix $P = D^{-1}W$.
Step 3: Compute conditioned constraint $V = P^T U$.
Step 4: Compute projected weight matrix $\tilde{W} = QP = P - D^{-1}V(V^T D^{-1}V)^{-1}V^T P$.
Step 5: Compute the second largest eigenvector $x^*$: $\tilde{W}x^* = \lambda x^*$.
Step 6: Threshold $x^*$ to get a discrete segmentation.

## 3   Results and conclusions

We apply our method to the images in Fig. 2. For all the examples, we compute pixel affinity $W$ as in Fig. 1. All the segmentation results are obtained by thresholding the eigenvectors using their mean values. The results without bias, with simple bias $U^T x = 0$ and conditioned bias $U^T Px = 0$ are compared in Fig. 5, 6, 7.

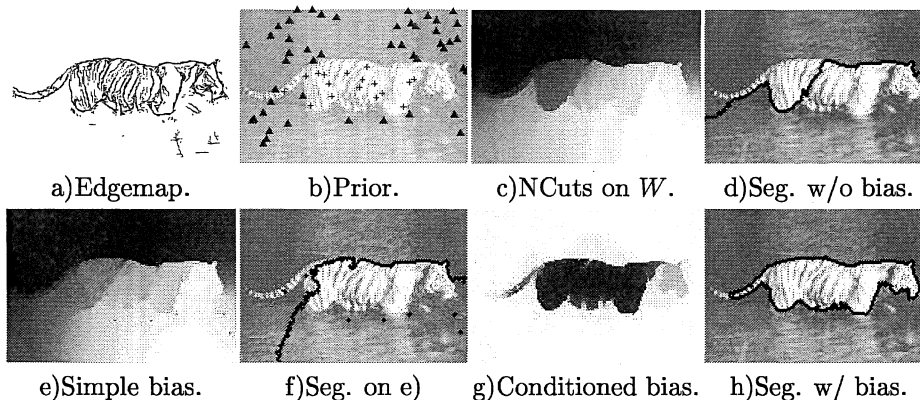

a)Edgemap.        b)Prior.        c)NCuts on $W$.        d)Seg. w/o bias.

e)Simple bias.        f)Seg. on e)        g)Conditioned bias.        h)Seg. w/ bias.

Figure 5: Segmentation with bias from unitary generative models. a)Edge map of the $100 \times 150$ image. $N = 15000$. b)We randomly sample 17 brightest pixels for $H_1$ (+), 48 darkest pixels for $H_2$ ($\blacktriangle$), producing 63 constraints in total. c) and d) show the solution without bias. It picks up the optimal bisection based on intensity distribution. e) and f) show the solution with simple bias. The labeled nodes have an uneven influence on grouping. g) and h) show the solution with conditioned bias. It successfully breaks the image into tiger and river as our general impression of this image. The computation for the three cases takes 11, 9 and 91ms respectively.

Prior knowledge is particularly useful in supplying long-range grouping information which often lacks in data grouping based on low level cues. With our model, the partial grouping prior can be integrated into the bottom-up grouping framework by seeking the optimal solution in a restricted domain. We show that the uniformity constraint is effective in eliminating spurious solutions resulting from simple perturbation on the optimal unbiased solutions. Segmentation from the discretization of the continuous eigenvectors also becomes trivial.

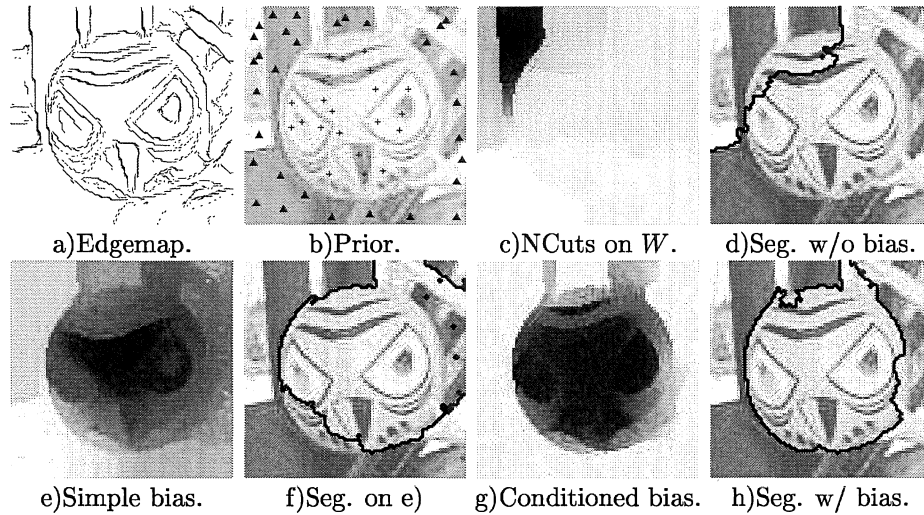

a)Edgemap.      b)Prior.      c)NCuts on $W$.    d)Seg. w/o bias.

e)Simple bias.    f)Seg. on e)    g)Conditioned bias.   h)Seg. w/ bias.

Figure 6: Segmentation with bias from hand-labeled partial grouping. a)Edge map of the $80 \times 82$ image. $N = 6560$. b)Hand-labeled partial grouping includes 21 pixels for $H_1$ (+), 31 pixels for $H_2$ (▲), producing 50 constraints in total. c) and d) show the solution without bias. It favors a few largest nearby pieces of similar intensity. e) and f) show the solution with simple bias. Labeled pixels in cluttered contexts are poor at binding their segments together. g) and h) show the solution with conditioned bias. It successfully pops out the pumpkin made of many small intensity patches. The computation for the three cases takes 5, 5 and 71ms respectively.

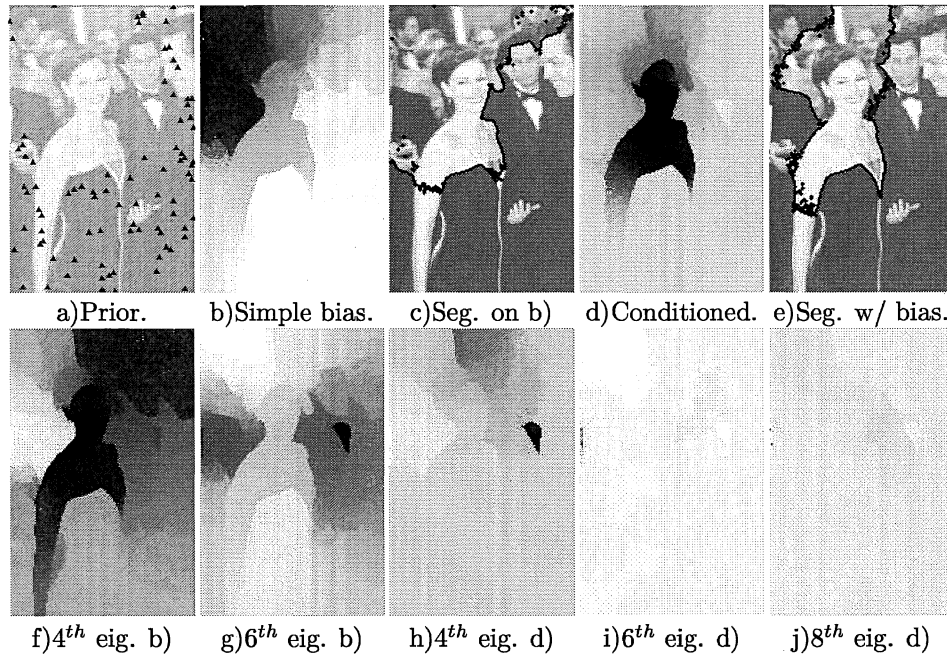

a)Prior.    b)Simple bias.    c)Seg. on b)   d)Conditioned.  e)Seg. w/ bias.

f)$4^{th}$ eig. b)    g)$6^{th}$ eig. b)    h)$4^{th}$ eig. d)    i)$6^{th}$ eig. d)    j)$8^{th}$ eig. d)

Figure 7: Segmentation with bias from spatial attention. $N = 25800$. a)We randomly choose 86 pixels far away from the fovea (Fig. 2c) for $H_2$ (▲), producing 85 constraints. b) and c) show the solution with simple bias. It is similar to the solution without bias (Fig. 1). d) and e) show the solution with conditioned bias. It ignores the variation in the background scene, which includes not only large pieces of constant intensity, but also many small segments of various intensities. The foreground successfully clips out the human figure. f) and g) are two subsequent eigenvectors with simple bias. h), i) and j) are those with conditioned bias. There is still a lot of structural organization in the former, but almost none in the latter. This greatly simplifies the task we face when seeking a segmentation from the continuous eigensolution. The computation for the three cases takes 16, 25 and 220ms respectively.

All these benefits come at a computational cost that is 10 times that for the original unbiased grouping problem. We note that we can also impose both $U^T x = 0$ and $U^T P x = 0$, or even $U^T P^s x = 0$, $s > 1$. Little improvement is observed in our examples. Since projected weight matrix $\tilde{W}$ becomes denser, the computation slows down. We hope that this problem can be alleviated by using multi-scale techniques. It remains open for future research.

### Acknowledgements

This research is supported by (DARPA HumanID) ONR N00014-00-1-0915 and NSF IRI-9817496.

## References

[1] A. Amir and M. Lindenbaum. Quantitative analysis of grouping process. In *European Conference on Computer Vision*, pages 371–84, 1996.

[2] A. Blum and S. Chawla. Learning from labeled and unlabeled data using graph mincuts, 2001.

[3] Y. Boykov, O. Veksler, and R. Zabih. Fast approximate energy minimization via graph cuts. In *International Conference on Computer Vision*, 1999.

[4] Y. Gdalyahu, D. Weinshall, and M. Werman. A randomized algorithm for pairwise clustering. In *Neural Information Processing Systems*, pages 424–30, 1998.

[5] S. Geman and D. Geman. Stochastic relaxation, Gibbs distributions, and the Bayesian restoration of images. *IEEE Transactions on Pattern Analysis and Machine Intelligence*, 6(6):721–41, 1984.

[6] H. Ishikawa and D. Geiger. Segmentation by grouping junctions. In *IEEE Conference on Computer Vision and Pattern Recognition*, 1998.

[7] I. H. Jermyn and H. Ishikawa. Globally optimal regions and boundaries. In *International Conference on Computer Vision*, 1999.

[8] M. Kass, A. Witkin, and D. Terzopoulos. Snakes: Active contour models. *International Journal of Computer Vision*, pages 321–331, 1988.

[9] J. Malik, S. Belongie, T. Leung, and J. Shi. Contour and texture analysis for image segmentation. *International Journal of Computer Vision*, 2001.

[10] M. Meila and J. Shi. Learning segmentation with random walk. In *Neural Information Processing Systems*, 2001.

[11] P. Perona and W. Freeman. A factorization approach to grouping. In *European Conference on Computer Vision*, pages 655–70, 1998.

[12] J. Puzicha, T. Hofmann, and J. Buhmann. Unsupervised texture segmentation in a deterministic annealing framework. *IEEE Transactions on Pattern Analysis and Machine Intelligence*, 20(8):803–18, 1998.

[13] S. Roy and I. J. Cox. A maximum-flow formulation of the $n$-camera stereo correspondence problem. In *International Conference on Computer Vision*, 1998.

[14] E. Sharon, A. Brandt, and R. Basri. Fast multiscale image segmentation. In *IEEE Conference on Computer Vision and Pattern Recognition*, pages 70–7, 2000.

[15] J. Shi and J. Malik. Normalized cuts and image segmentation. In *IEEE Conference on Computer Vision and Pattern Recognition*, pages 731–7, June 1997.
